# Limiting form of the sample covariance eigenspectrum in PCA and kernel PCA

**David C. Hoyle & Magnus Rattray**
Department of Computer Science,
University of Manchester,
Manchester M13 9PL, UK.
david.c.hoyle@man.ac.uk   magnus@cs.man.ac.uk

## Abstract

We derive the limiting form of the eigenvalue spectrum for sample co-variance matrices produced from non-isotropic data. For the analysis of standard PCA we study the case where the data has increased variance along a small number of symmetry-breaking directions. The spectrum depends on the strength of the symmetry-breaking signals and on a parameter $\alpha$ which is the ratio of sample size to data dimension. Results are derived in the limit of large data dimension while keeping $\alpha$ fixed. As $\alpha$ increases there are transitions in which delta functions emerge from the upper end of the bulk spectrum, corresponding to the symmetry-breaking directions in the data, and we calculate the bias in the corresponding eigenvalues. For kernel PCA the covariance matrix in feature space may contain symmetry-breaking structure even when the data components are independently distributed with equal variance. We show examples of phase-transition behaviour analogous to the PCA results in this case.

## 1   Introduction

A number of data analysis methods are based on the spectral decomposition of large matrices. Examples include Principal Component Analysis (PCA), kernel PCA and spectral clustering methods. PCA in particular is a ubiquitous method of data analysis [1]. The principal components are eigenvectors of the sample covariance matrix ordered according to the size of the corresponding eigenvalues. In PCA the data is projected onto the subspace corresponding to the $n$ first principal components, where $n$ is chosen according to some model selection criterion. Most methods for model selection require only the eigenvalue spectrum of the sample covariance matrix. It is therefore useful to understand how the sample covariance spectrum behaves given a particular data distribution. Much is known about the asymptotic properties of the spectrum in the case where the data distribution is isotropic, e.g. for the Gaussian Orthogonal Ensemble (GOE), and this knowledge can be used to construct model selection methods (see e.g. [2] and references therein). However, it is also instructive to consider the limiting behaviour in the case where the data does contain some low-dimensional structure. This is interesting as it allows us to understand the limits of learnability and previous studies have already shown phase-transition behaviour in PCA learning from data containing a single symmetry-breaking direction [3]. The analysis of

data models which include a signal component are also useful if we are to correct for bias in the estimated eigenvalues corresponding to retained components.

PCA has limited applicability because it is a globally linear method. A promising non-linear alternative is kernel PCA [4] in which data is projected into a high-dimensional feature space and PCA is carried out in this feature space. The kernel trick allows all computations to be carried out efficiently so that the method is practical even when the feature space has a very high, or even infinite, dimension. In this case we are interested in properties of the eigenvalue spectrum of the sample covariance matrix in feature space. The covariance of the features will typically be non-isotropic even when the data itself has independently distributed components with equal variance. The sample covariance spectrum will therefore show quite rich behaviour even when the data itself has no structure. It is important to understand the expected behaviour in order to develop model selection methods for kernel PCA analogous to those used for standard PCA. Model selection methods based on data models with isotropic noise (e.g. [2, 5]) are certainly not suitable for kernel PCA.

In this paper we apply methods from statistical mechanics and random matrix theory to determine the limiting form of eigenvalue spectrum for sample covariance matrices produced from data containing symmetry-breaking structure. We first show how the replica method can be used to derive the spectrum for Gaussian data with a finite number a symmetry-breaking directions. This result is confirmed and generalised by studying the Stieltjes transform of the eigenvalue spectrum, suggesting that it may be insensitive to details of the data distribution. We then show how the results can be used to derive the limiting form of eigenvalue spectrum of the feature covariance matrix (or Gram matrix) in kernel PCA for the case of a polynomial kernel.

## 2    Statistical mechanics theory for Gaussian data

We first consider a data set of $N$-dimensional data vectors $\{\boldsymbol{x}_\mu\}_{\mu=1}^p$ containing a signal and noise component. Initially we restrict ourselves to the case where $\boldsymbol{x}_\mu$ is drawn from a Gaussian distribution whose covariance matrix $\boldsymbol{C}$ is isotropic except for a small number of orthogonal symmetry-breaking directions, i.e.,

$$\boldsymbol{C} \;=\; \sigma^2 \boldsymbol{I} \;+\; \sigma^2 \sum_{m=1}^{S} A_m \boldsymbol{B}_m \boldsymbol{B}_m^T \,, \quad \boldsymbol{B}_n^T \boldsymbol{B}_m = \delta_{nm} \,, \quad A_m > 0 \,. \tag{1}$$

We define the sample covariance $\hat{\boldsymbol{C}} = p^{-1} \sum_\mu \boldsymbol{x}_\mu \boldsymbol{x}_\mu^T$ and study its eigenvalue spectrum in the limit $N \to \infty$ when the ratio $\alpha = p/N$ is held fixed and the number of symmetry-breaking directions $S$ is finite. We work with the trace of the resolvent $\boldsymbol{G}(\lambda) = (\lambda \boldsymbol{I} - \hat{\boldsymbol{C}})^{-1}$ from which the density of eigenvalues $\rho(\lambda)$ can be calculated,

$$\rho(\lambda) = \lim_{\epsilon \to 0^+} (N\pi)^{-1} \mathrm{Im}\, \mathrm{tr} \boldsymbol{G}(\lambda - i\epsilon) \quad \text{where} \quad \mathrm{tr} \boldsymbol{G}(\lambda) = \sum_{i=1}^{N} \frac{1}{\lambda - \lambda_i} \tag{2}$$

and $\lambda_i$ are eigenvalues of $\hat{\boldsymbol{C}}$. The trace of the resolvent can be represented as,

$$\mathrm{tr} \boldsymbol{G}(\lambda) \;=\; \frac{\partial}{\partial \lambda} \log \det(\lambda \boldsymbol{I} - \hat{\boldsymbol{C}}) \;=\; \frac{\partial}{\partial \lambda} \log Z(\lambda) \,. \tag{3}$$

Using the standard representation of the determinant of a matrix,

$$[\det \boldsymbol{A}]^{-\frac{1}{2}} \;=\; (2\pi)^{-\frac{N}{2}} \int \exp\left[-\tfrac{1}{2}\boldsymbol{\phi}^T \boldsymbol{A} \boldsymbol{\phi}\right] d\boldsymbol{\phi} \,,$$

we have,

$$\log Z(\lambda) \;=\; N \log 2\pi \;-\; 2 \log \int \exp\left[-\frac{\lambda}{2}||\boldsymbol{\phi}||^2 \;+\; \frac{1}{2p} \sum_\mu (\boldsymbol{\phi} \cdot \boldsymbol{x}_\mu)^2\right] d\boldsymbol{\phi} \,. \tag{4}$$

We assume that the eigenvalue spectrum is self-averaging, so that the calculation for a specific realisation of the sample covariance can be replaced by an ensemble average for large $N$ that can be performed using the replica method (see e.g. [6]). Details are presented elsewhere [7] and here we simply state the results. The calculation is similar to [3] where Reimann *et. al.* study the performance of PCA on Gaussian data with a single symmetry-breaking direction, although there are also notable differences between the calculations.

We find the following asymptotic result for the spectral density,

$$
\begin{aligned}
\rho(\lambda) \quad = \quad & (1-\alpha)\Theta(1-\alpha)\delta(\lambda) \;+\; \frac{1}{N}\sum_{m=1}^{S}\delta(\lambda - \lambda_u(A_m, \sigma^2))\Theta(\alpha - A_m^{-2}) \\
+ \quad & \left(1 - \frac{1}{N}\sum_{m=1}^{S}\Theta(\alpha - A_m^{-2})\right)\frac{\alpha}{2\pi\lambda\sigma^2}\sqrt{\mathrm{Max}(0, (\lambda - \lambda_{\min})(\lambda_{\max} - \lambda))}\,, \quad (5)
\end{aligned}
$$

where we have defined,

$$
\lambda_{\max,\min} = \sigma^2\alpha^{-1}(1 \pm \sqrt{\alpha})^2 \qquad \lambda_u(A, \sigma^2) = \sigma^2(1 + A)(1 + \frac{1}{\alpha A})\,. \quad (6)
$$

The first term in equation (5) sets a proportion $1 - \alpha$ eigenvalues to zero when the rank of $\hat{C}$ is less than $N$, i.e. when $\alpha < 1$. The last term represents the bulk of the spectrum and is identical to the well-known Marčenko-Pastur law for isotropic data with variance $\sigma^2$ [8, 9]. In [7] we also give the $O(1/N)$ corrections to this term, but here we are mainly interested in the leading order. The second term contains contributions due to the underlying structure in the data. The $m$th symmetry-breaking term in the data covariance $C$ only contributes to the spectrum if $\alpha > A_m^{-2}$. This transition must be exceeded before signals of a given strength can be detected, i.e. the signal must be sufficiently strong or the data set sufficiently large. This corresponds to the same learning transition point observed in studies of PCA on Gaussian data with a single symmetry-breaking direction [3]. Above this transition the sample covariance eigenvalue over-estimates the true variance corresponding to this component by a factor $1 + 1/(\alpha A_m)$ which indicates a significant bias when the data set is small or the signal is relatively weak. Our result provides a method of bias correction for the top eigenvalues in this case.

In figure 1 we show results for Gaussian data with three symmetry-breaking directions, each above the transition point. On the left we show how the top eigenvalues separate from the bulk while the inset compares the density of the bulk with the theoretical result, showing excellent agreement. On the right we show convergence to the theoretical result for $\lambda_u(A, \sigma^2)$ in equation (6) as the data dimension $N$ is increased for fixed $\alpha$.

## 3 Analysis of the Stieltjes transform

The statistical mechanics approach is useful because it allows the derivation of results from first principles and it is possible to use this method to determine other self-averaging quantities of interest, e.g. the overlap between the leading eigenvectors of the sample and population covariances [3]. However, the method as presented here is restricted to Gaussian data. A number of results from the statistics literature have been derived under much weaker and often more explicit assumptions about the data distribution. It is therefore interesting to ask whether equation (5) can also be derived from these results.

Marčenko and Pastur [8] studied the case of data with a general covariance matrix. The limiting distribution was shown to satisfy,

$$
\rho(\lambda) \quad = \quad \lim_{\epsilon \to 0^+} \pi^{-1}\mathrm{Im}\,\alpha m_\rho(\lambda + i\epsilon) \quad \text{where} \quad m_\rho(z) = \alpha^{-1}\int \mathrm{d}\lambda\,\frac{\rho(\lambda)}{\lambda - z}\,. \quad (7)
$$

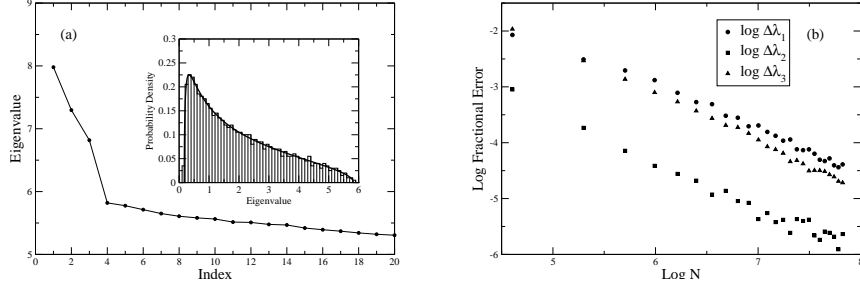

Figure 1: In (a) we show eigenvalues of the sample covariance matrix for Gaussian data with $\sigma^2 = 1$, $N = 2000$ and $\alpha = 0.5$. The data contains three symmetry-breaking directions with strengths $A_1^2 = 20$, $A_2^2 = 15$ and $A_3^2 = 10$ all above the transition point. The inset shows the distribution of all non-zero eigenvalues except for the largest three with the solid line showing the theoretical result. In (b) we show the fractional difference between the three largest eigenvalues $\lambda_i$ and the theoretical value $\lambda_u(A_i, \sigma^2)$ for $i = 1, 2, 3$. We set $\alpha = 0.2$, averaged $\lambda_i$ over 1000 samples to get $\langle\lambda_i\rangle$, set $\Delta\lambda_i = |1 - \langle\lambda_i\rangle/\lambda_u(A_i, \sigma^2))|$ and set other values as in (a).

Here, $m_\rho(z)$ is the Stieltjes transform of $\alpha^{-1}\rho(\lambda)$ and is equal to $-p^{-1}\mathrm{trG}(z)$. The above equation is therefore exactly equivalent to equation (2) and we see that this approach starts from the same point as the statistical mechanics theory. Marčenko and Pastur showed that the Stieltjes transform satisfies the following relationship,

$$z(m_\rho) = -\frac{1}{m_\rho} + \alpha^{-1}\int \frac{dH(t)}{t^{-1} + m_\rho} . \tag{8}$$

The measure $H(t)$ is defined such that $N^{-1}\sum_i d_i^k$ converges to $\int t^k dH(t)$ $\forall k$ where $d_i$ are the eigenvalues of $\boldsymbol{C}$. An equivalent result is also derived by Wachter [10] and more recently by Sengupta and Mitra using the replica method [11] (for Gaussian data). Silverstein and Choi have shown that the support of $\rho(\lambda)$ can be determined by the intervals between extrema of $z(m_\rho)$ [12] and this has been used to determine the signal component of a spectrum when $O(N)$ equal strength symmetry-breaking directions are present [13].

Since $\boldsymbol{C}$ in equation (1) only contains a finite number of symmetry-breaking directions then in the limit $N \to \infty$ these will have zero measure as defined by $H$. Thus, in this limit the eigenvalue density would appear to be identical to the isotropic case. However, it is the behaviour of the largest eigenvalues that we are most interested in, even though these may have vanishing measure. For the case of a single symmetry-breaking direction ($S = 1$, $A_1 = A$) we take $dH(t) = (1 - \epsilon)\delta(t - \sigma^2)dt + \epsilon\delta(t - \sigma^2(1 + A))dt$, with $\epsilon \simeq 1/N$. This gives,

$$z(m_\rho) = -\frac{1}{m_\rho} + \frac{(1 - \epsilon)\alpha^{-1}}{\sigma^{-2} + m_\rho} + \frac{\epsilon\alpha^{-1}}{\sigma^{-2}(1 + A)^{-1} + m_\rho} , \tag{9}$$

and stationary points satisfy,

$$0 = \frac{1}{m_\rho^2} - \frac{(1 - \epsilon)\alpha^{-1}}{(\sigma^{-2} + m_\rho)^2} - \frac{\epsilon\alpha^{-1}}{(\sigma^{-2}(1 + A)^{-1} + m_\rho)^2} . \tag{10}$$

Since $\epsilon \ll 1$ we do not expect the behaviour of $z(m_\rho)$ to be modified substantially in the interval $[\lambda_{\min}, \lambda_{\max}]$. Therefore we look for additional stationary points close to the singularity at $m_\rho = -\sigma^{-2}(1+A)^{-1}$. Setting $m_\rho = -\sigma^{-2}(1+A)^{-1} + \delta$ and expanding (10)

yields $\delta = \epsilon^{\frac{1}{2}}/\sigma^2(1+A)\sqrt{(\alpha - A^{-2})} + \mathcal{O}(\epsilon)$. Substituting this into (9) gives $z(-\sigma^{-2}(1+A)^{-1}+\delta) = \sigma^2(1+A)(1+(\alpha A)^{-1}) + \mathcal{O}(\epsilon^{\frac{1}{2}})$. Thus, as $N \to \infty$, if the stationary points at $-\sigma^{-2}(1+A)^{-1}+\delta$ exist they will define a small interval of $z$ centred on $\lambda_u(A, \sigma^2)$ and so define an approximate contribution of $N^{-1}\delta(\lambda - \lambda_u(A, \sigma^2))$ to the spectrum, in agreement with the previous calculations using replicas. We also see that for $\delta$ to be real requires $\alpha > A^{-2}$, in agreement with our previous calculation for the learning transition point. A similar perturbative analysis when $C$ contains more than one symmetry-breaking direction gives a set of contributions $N^{-1}\delta(\lambda - \lambda_u(A_m, \sigma^2))$, $m = 1, \ldots, S$, to $\rho(\lambda)$. Again this is in agreement with our previous replica analysis of the resolvent.

The relationship in equation (8) can be obtained with only relatively weak conditions on the data distribution. One requirement is that the second moment of each element of $\hat{C}$ exists. Bai has considered the case of data vectors with non-Gaussian *i.i.d.* components (e.g. [14]) while Marčenko and Pastur show that the data vector components do not have to be independently distributed for the relation to hold and they give sufficient conditions on the 4th order cross-moments of the data vector components [8]. In [7] we study PCA on some examples of non-Gaussian data with symmetry-breaking structure (non-Gaussian signal and noise) and show that the separated eigenvalues behave similarly to figure 1.

## 4 Eigenvalue spectra for kernel PCA

Equation (8) holds under quite weak conditions on the data distribution. It is therefore hoped that we can apply these results to the feature space of kernel PCA [4]. In kernel PCA the data $x$ is transformed into a feature vector $\phi(x)$ and standard PCA is carried out in the feature space. The method requires that we can define a kernel function $k(x, y) = \phi(x) \cdot \phi(y)$ that allows efficient computation of the dot-product in a high, or even infinite, dimensional space. The eigenvalues of the sample covariance in feature space are identical to eigenvalues of the Gram matrix $K_{\mu\nu}$ with entries $k(x_\mu, x_\nu)$ and the eigenvalues can therefore be computed efficiently for arbitrary feature-space dimension as long as the number of samples $p$ is not too large (NB. The Gram matrix first has to be centred [4] so that the data has zero mean in the feature space).

One common choice of kernel function is the polynomial kernel $k(x, y) = (c + x \cdot y)^d$ in which case, for integer $d$, the features are all possible monomials up to order $d$ involving components of $x$. We limit our attention here to the quadratic kernel ($d = 2$). We consider data vectors with components that are independently and symmetrically distributed with equal variance $\sigma^2$ and choose a set of features $\phi(x) = (\sqrt{2c}\,x, \text{Vec}[xx^\mathrm{T}])$ where $\text{Vec}[xx^\mathrm{T}]_{j+N(i-1)} = x_i x_j$. The covariance in feature space is block diagonal,

$$C = \left( \begin{array}{c|c} 2c\langle xx^\mathrm{T}\rangle & \mathbf{0} \\ \hline \mathbf{0} & \langle\text{Vec}[xx^\mathrm{T}]\text{Vec}[xx^\mathrm{T}]^\mathrm{T}\rangle \\ & -\langle\text{Vec}[xx^\mathrm{T}]\rangle\langle\text{Vec}[xx^\mathrm{T}]^\mathrm{T}\rangle \end{array} \right)$$

| $d_i$ | number |
|---|---|
| $2c\sigma^2$ | $N$ |
| $2\sigma^4$ | $N(N-1)/2$ |
| $2\sigma^4 + \kappa_4^i$ | $N$ |

where angled brackets denote expectations over the data distribution. The non-zero eigenvalues of $C$ are shown on the right where $\kappa_4^i = \langle x_i^4\rangle - 3\sigma^4$ is the 4th cumulant of the $i$th component of $x$. We see that although each component of the data is independently distributed with equal variance, the covariance structure in feature space may be quite complex.

#### • Gaussian data, $c = 0$

For isotropic Gaussian data and $c = 0$ there is a single degenerate eigenvalue of $C$ and the asymptotic result for the spectrum is identical to the case of an isotropic distribution [8, 9] with variance $2\sigma^4$ and $\alpha$ defined as the ratio of the number of examples $p$ to the effective

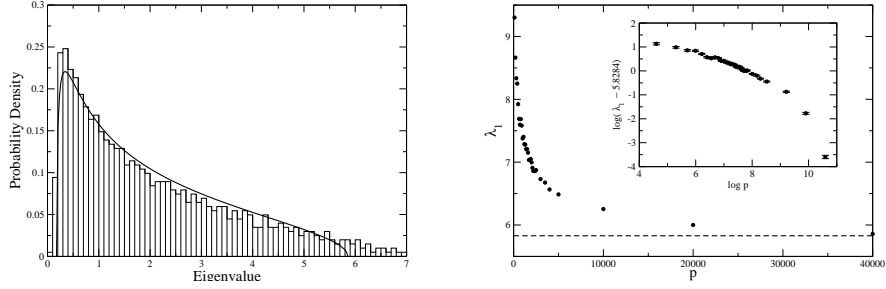

Figure 2: On the left we show the Gram matrix eigenspectrum for a sample data set and compare it to the theoretical result. The kernel is purely quadratic ($c = 0$) and we use isotropic Gaussian data with $2\sigma^4 = 1$, $N = 63$ and $p = 1000$ so that $\alpha \simeq 0.5$. On the right we show the averaged top eigenvalue against $p$ for fixed $\alpha$. Each point is averaged over 100 samples except for the right-most which is averaged over 50. The dashed line shows the theoretical result $\lambda_1 = 5.8284$ and inset is a log-log plot of the same data.

dimension in the feature space $N(N+1)/2$ (i.e. the degeneracy of the non-zero eigenvalue) so that $\alpha = 2p/N(N + 1)$ and $p = O(N^2)$ is the appropriate scaling.

On the left of figure 2 we compare the spectra for a single sample data set to the theory for $p = 1000$ and $N = 63$ which corresponds to $\alpha \simeq 0.50$ and the theoretical curve is almost identical to the one used in the inset to figure 1(a). The finite size effects are much larger than would be observed for PCA with isotropic data and on the right of figure 2 we show the average of the top eigenvalue for this value of $\alpha$ as $p$ is increased, showing a very slow convergence to the asymptotic result.

• **Gaussian data, $c > 0$**

For isotropic Gaussian data and $c > 0$ there are two eigenvalues of $C$ with degeneracy $N$ and $N(N + 1)/2$ respectively. For large $N$ and $c > \sigma^2$ the top $N$ eigenvalues play an analogous role to the top $S$ eigenvalues in the PCA data model defined in section 2. A similar perturbative expansion to the one described in section 3 shows that when $\alpha < (c/\sigma^2 - 1)^{-2}$ (where $\alpha \simeq 2p/N^2$ is defined relative to the feature space) the distribution is identical to the $c = 0$ case. For $\alpha$ above this transition point the $N$ top eigenvalues separate from the bulk. In the limit $N \to \infty$ with $p = O(N^2)$ the spread of the upper $N$ eigenvalues will tend to zero and they will become localised at $\lambda_u(c/\sigma^2 - 1, 2\sigma^4)$ as defined by equation (6). For finite $N$ and when the two components of the spectra are well separated, we can approximate the eigenvalue spectrum of the top $N$ eigenvalues as though the data only contains these components, i.e. we model this cluster as isotropic data with $\alpha = p/N$ and variance $2c\sigma^2$. We obtain an improved approximation by correcting the mean of the separated cluster by the value predicted for the mean in the large $N$ limit.

On the left of figure 3 we compare this approximation to the Gram matrix spectrum averaged over 300 data sets for large $c$, with the inset showing the separated cluster. The theory is shown by the solid line and provides a good qualitative fit to the data although there are significant discrepancies. For the bulk we believe these to be due to finite size effects but the theory for the spread of the upper $N$ eigenvalues is only approximate since the spread of this cluster will vanish as $N \to \infty$ for fixed $c$ and $p = O(N^2)$. On the right of figure 3 we plot the average of the top $N$ eigenvalues against $c$, showing good agreement with the theory. The top eigenvalue of the population covariance is shown by the line and the theory accurately predicts the bias in the sample estimate.

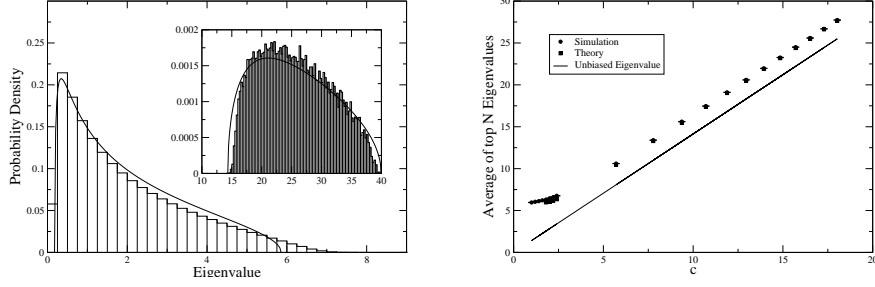

Figure 3: On the left we show the Gram matrix eigenvalue spectrum averaged over 300 data sets and compare it to the theoretical result. The inset shows the density of the top $N$ eigenvalues which are separated from the bulk. The kernel is quadratic with $c = \sigma^2(1 + \sqrt{500})$ with other parameters as in figure 2. On the right we show the average of the top $N$ eigenvalues against the theoretical result as a function of $c$.

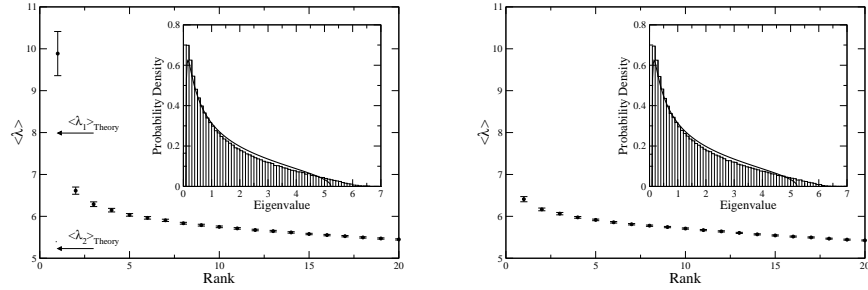

Figure 4: Results from a purely quadratic kernel ($c = 0$) on data containing a single dimension having positive kurtosis. We show the top 20 eigenvalues of the Gram matrix with the bulk spectrum as an inset. On the left $\kappa_4 = 5$ and we are above the transition where the top eigenvalue is separated from the bulk. On the right $\kappa_4 = 1$ is below the transition. Other parameters were $2\sigma^4 = 1$, $N = 70$, $p = 1500$ and results were averaged over 25 data sets.

• **Non-Gaussian data, $c = 0$**

If the data has components with positive kurtosis then these will break the symmetry of the covariance. This is analogous to the case for PCA studied in section 2 and the result for the limiting spectrum carries over. We have $\alpha \simeq 2p/N^2$ defined with respect to the dimension of the feature space. For each component of the data with $\kappa_4^i > 2\sigma^4/\sqrt{\alpha}$ there will be a delta function in the spectrum at $\lambda_u(\kappa_4^i/2\sigma^4, 2\sigma^4)$ as defined by equation (6).

In figure 4 we show the Gram matrix eigenvalues for a data set containing a single dimension having positive kurtosis. On the left we have $\kappa_4 = 5$ which is above the transition. We have indicated with arrows the theoretical prediction for the top two eigenvalues and we see that there is a significant difference, although the separation is quite well described by the theory. We expect that these discrepancies are due to large finite size effects and further simulations are required to verify this. On the right we have $\kappa_4 = 1$ which is below the transition and the spectrum is very similar to the case for isotropic Gaussian data.

# 5   Conclusion

We studied the asymptotic form of the sample covariance eigenvalue spectrum from data with symmetry-breaking structure. For standard PCA the asymptotic results are very accurate even for moderate data dimension, but for kernel PCA with a quadratic kernel we found that convergence to the asymptotic result was slow. The limiting form of sample covariance spectra has previously been studied in the neural networks literature where it can be used in order to determine the optimal batch learning rate for large linear perceptrons. Indeed, the results derived in section 2 for Gaussian data can also be derived by adapting an elegant method developed by Sollich [15], without recourse to the replica method. Halkjær & Winther used this approach to compute the spectral density for the case of a single symmetry breaking direction and obtained a similar result to us, except that the position of the separated eigenvalue was at $\sigma^2(1 + A)$ which differs from our result [16]. In fact they assumed a large signal in their derivation and their derivation can easily be adapted to obtain an identical result to ours. However this method, as well as the replica approach used here, is limited because it only applies to Gaussian data, while the Stieltjes transform relationship in equation (8) has been derived under much weaker conditions on the data distribution.

Our current work is focussed on extending the analysis to more general kernels, such as the radial basis function (RBF) kernel where the feature space dimension is infinite. In the general case we find that the Stieltjes transform can be derived by a variational mean field theory and therefore provides a principled approximation to the average spectral density.

**Acknowledgments** DCH was supported by a MRC(UK) Special Training Fellowship in Bioinformatics. We would like to thank the anonymous reviewers for useful comments and for pointing out references [15] and [16].

# References

[1] I.T. Jolliffe. *Principal Component Analysis*. Springer-Verlag, New York, 1986.

[2] I.M. Johnstone. *Ann. Stat.*, 29, 2001.

[3] P. Reimann, C. Van den Broeck, and G.J. Bex. *J. Phys. A:Math. Gen.*, 29:3521, 1996.

[4] B. Scholköpf, A. Smola, and K.-R. Müller. *Neural Computation*, 10:1299–1319, 1998.

[5] T.P. Minka. Automatic choice of dimensionality for PCA. In T.K. Leen, T.G. Dietterich, and V. Tresp, editors, *NIPS 13*, pages 598–604. MIT Press, 2001.

[6] A. Engel and C. Van den Broeck. *Statistical Mechanics of Learning*. Cambridge University Press, 2001.

[7] D.C. Hoyle and M. Rattray. *Phys. Rev. E*, in press.

[8] V.A. Marčenko and L.A. Pastur. *Math. USSR-Sb*, 1:507, 1967.

[9] A. Edelman. *SIAM J. Matrix Anal. Appl.*, 9:543, 1988.

[10] K.W. Wachter. *Ann. Probab.*, 6:1, 1978.

[11] A.M. Sengupta and P.P. Mitra. *Phys. Rev. E*, 60:3389, 1999.

[12] J.W. Silverstein and S. Choi. *J. Multivariate Analysis*, 54:295, 1995.

[13] J.W. Silverstein and P.L. Combettes. *IEEE Trans. Signal Processing*, 40:2100, 1992.

[14] Z.D. Bai. *Ann. Probab.*, 21:649, 1993.

[15] P. Sollich. *J. Phys. A*, 27:7771, 1994.

[16] S. Halkjær and O. Winther. In M. Mozer, M. Jordan, and T. Petsche, editors, *NIPS 9*, page 169. MIT Press, 1997.
